# Online Sum-Product Computation over Trees

**Mark Herbster**     **Stephen Pasteris**
Department of Computer Science
University College London
London WC1E 6BT, England, UK
{m.herbster, s.pasteris}@cs.ucl.ac.uk

**Fabio Vitale**
Department of Computer Science
University of Milan
20135 Milan, Italy
fabio.vitale@unimi.it

## Abstract

We consider the problem of performing efficient sum-product computations in an online setting over a tree. A natural application of our methods is to compute the marginal distribution at a vertex in a tree-structured Markov random field. Belief propagation can be used to solve this problem, but requires time linear in the size of the tree, and is therefore too slow in an online setting where we are continuously receiving new data and computing individual marginals. With our method we aim to update the data and compute marginals in time that is no more than logarithmic in the size of the tree, and is often significantly less. We accomplish this via a hierarchical covering structure that caches previous local sum-product computations. Our contribution is three-fold: we i) give a linear time algorithm to find an optimal hierarchical cover of a tree; ii) give a sum-product-like algorithm to efficiently compute marginals with respect to this cover; and iii) apply "i" and "ii" to find an efficient algorithm with a regret bound for the online *allocation* problem in a multi-task setting.

## 1   Introduction

The use of graphical models [1, 2] is ubiquitous in machine learning. The application of the batch sum-product algorithm to tree-structured graphical models, including hidden Markov models, Kalman filtering and turbo decoding, is surveyed in [3]. Our aim is to adapt these techniques to an online setting.

In our online model we are given a tree and a fixed set of parameters. We then receive a potentially unbounded online sequence of "prediction requests" and "data updates." A prediction request indicates a vertex for which we then return the posterior marginal at that vertex. Each data update associates a new "factor" to that vertex. Classical belief propagation requires time linear in the size of the tree for this task. Our algorithm requires time linear in the height of an optimal hierarchical cover of this tree. The height of the cover is in the worst case logarithmic in the size the tree. Thus our per trial prediction/update time is at least an exponential improvement over classical belief propagation.

The paper is structured as follows. In Section 2 we introduce our notation leading to our definition of an optimal hierarchical cover. In Section 3 we give our optimal hierarchical covering algorithm. In Section 4 we show how we may use this cover as a structure to cache computations in our sum-product-like algorithm. Finally, in Section 5 we give a regret bound and a sketch of an application of our techniques to an online multi-task *allocation* [4] problem.

**Previous work.** Pearl [5] introduced belief propagation for Bayes nets which computes marginals in time linear in the size of the tree. In [6] an algorithm for the online setting was given for a Bayes net on a tree $T$ which required $\mathcal{O}(\log |V(T)|)$ time per marginalization step, where $|V(T)|$ is the number of vertices in the tree. In this work we consider a Markov random field on a tree. We give an algorithm whose performance is bounded by $\mathcal{O}(\chi^*(T))$. The term $\chi^*(T)$ is the height of our

optimal hierarchical cover which is upper bounded by $\mathcal{O}(\min(\log |V(T)|, \text{diameter}(T)))$, but may in fact be exponentially smaller.

## 2 Hierarchical cover of a tree

In this section we introduce our notion of a **hierarchical cover** of a tree and its dual the **decomposition tree**.

**Graph-theoretical preliminaries.** A graph $G$ is a pair of sets $(V, E)$ such that $E$ is a set of unordered pairs of distinct elements from $V$. The elements of $V$ are called vertices and those of $E$ are called edges. In order to avoid ambiguities deriving from dealing with different graphs, in some cases we will highlight the membership to graph $G$ denoting these sets as $V(G)$ and $E(G)$ respectively. With slight abuse of notation, by writing $v \in G$, we mean $v \in V(G)$. $S$ is a subgraph $G$ (we write $S \subseteq G$) iff $V(S) \subseteq V(G)$ and $E(S) = \{(i, j) : i, j \in V(S), (i, j) \in E(G)\}$. Given any subgraph $S \subseteq G$, we define its **boundary** (or inner border) $\partial_G(S)$ and its **neighbourhood** (or outer border) $N_G(S)$ as: $\partial_G(S) := \{i : i \in S, j \notin S, (i, j) \in E(G)\}$, and $N_G(S) := \{j : i \in S, j \notin S, (i, j) \in E(G)\}$. With slight abuse of notation, $N_G(v) := N_G(\{v\})$, and thus the degree of a vertex $v$ is $|N_G(v)|$. Given any graph $G$, we define the set of its leaves as $\text{leaves}(G) := \{i \in G : |N_G(i)| = 1\}$, and its **interior** $G^{\bullet} := \{i \in G : |N_G(i)| \neq 1\}$. A path $P$ in a graph $G$ is defined by a sequence of distinct vertices $\langle v_1, v_2, ..., v_m \rangle$ of $G$, such that for all $i < m$ we have that $(v_i, v_{i+1}) \in E(G)$. In this case we say that $v_1$ and $v_m$ are connected by the subgraph $P$. A tree $T$ is a graph in which for all $v, w \in T$ there exists a unique path connecting $v$ with $w$. In this paper we will only consider trees with a non-empty edge set and thus the vertex set will always have a cardinality of at least 2. The distance $d_T(v, w)$ between $v, w \in T$ is the path length $|E(P)|$. The pair $(T, r)$ denotes a **rooted** tree $T$ with root vertex $r$. Given a rooted tree $(T, r)$ and any vertex $i \in V(T)$, the (*proper*) descendants of $i$ are all vertices that can be connected with $r$ via paths $P \subseteq T$ containing $i$ (excluding $i$). Analogously, the (*proper*) ancestors of $i$ are all vertices that lie on the path $P \subseteq T$ connecting $i$ with $r$ (excluding $i$). We denote the set of all descendants (resp. all ancestors) of $i$ by $\Downarrow_T^r(i)$ (resp. $\Uparrow_T^r(i)$). We shall omit the root $r$ when it is clear from the context. Vertex $i$ is the parent (resp. child) of $j$, which is denoted by $\uparrow_T^r(j)$ (resp. $i \in \downarrow_T^r(j)$) if $(i, j) \in E(T)$ and $i \in \Uparrow_T^r(j)$ (resp. $i \in \Downarrow_T^r(j)$). Given a tree $T$ we use the notation $S \subseteq T$ *only* if $S$ is a tree and subgraph of $T$. The height of a rooted tree $(T, r)$ is the maximum length of a path $P \subseteq T$ connecting the root to any vertex: $h_r(T) := \max_{v \in T} d_T(v, r)$. The diameter $\Delta(T)$ of a tree $T$ is defined as the length of the longest path between any two vertices in $T$.

### 2.1 The hierarchical cover of a tree

In this section we describe a splitting process that recursively decomposes a given tree $T$. A (decomposition) tree $(D, r)$ identifies this splitting process, generating a tree-structured collection $\mathcal{S}$ of subtrees that hierarchically cover the given tree $T$.

This process recursively splits at each step a subtree of $T$ (that we call a "component") resulting from some previous splits. More precisely, a subtree $S \subseteq T$ is split into two or more subcomponents and the decomposition of $S$ depends only on the choice of a vertex $v \in S^{\bullet}$, which we call **splitting vertex**, in the following way. The splitting vertex $v \in S^{\bullet}$ of $S$ induces the **split** set $\Omega(S, v) = \{S_1, \ldots, S_{|N_S(v)|}\}$ which is the unique set of $S$'s subtrees which overlap at a vertex $v$, uniquely, that represent a cover for $S$, i.e., it satisfies (i) $\cup_{S' \in \Omega(S, v)} S' = S$ and (ii) $\{v\} = S_i \cap S_j$ for all $1 \leq i < j \leq |N_S(v)|$. Thus the split may be visualized by considering the forest $F$ resulting from removing a vertex from $S$, but afterwards each component $S_1, \ldots, S_{|N_S(v)|}$ of $F$ has the "removed vertex" $v$ added back to it. A component having only two vertices is called **atomic**, since it cannot be split further. We indicate with $S^v \subseteq T$ the component subtree whose splitting vertex is $v$, and we denote atomic components by $S^{(i,j)}$, where $E(S^{(i,j)}) = \{(i, j)\}$. We finally denote by $\mathcal{S}$ the set of all component subtrees obtained by this splitting process. Since the method is recursive, we can associate a rooted tree $(D, r)$, with $T$'s decomposition into a hierarchical cover, whose internal vertices are the splitting vertices of the splitting process. Its leaves correspond to the single edges (of $E(T)$) of each atomic component, and a vertex "parent-child" relation $c \in \downarrow_D^r(p)$ corresponds to the "splits-into" relation $S^c \in \Omega(S^p, p)$ (see Figure 1).

We will now formalize the splitting process by defining the **hierarchical cover** $\mathcal{S}$ of a tree $T$, which is a key concept used by our algorithm.

**Definition 1.** A ***hierarchical cover*** $\mathcal{S}$ *of a tree* $T$ *is a tree-structured collection of subtrees that hierarchically cover the tree* $T$ *satisfying the following three properties:*

1. $T \in \mathcal{S}$,

2. *for all* $S \in \mathcal{S}$ *with* $S^\bullet \neq \emptyset$ *there exists an* $x \in S^\bullet$ *such that* $\Omega(S, x) \subset \mathcal{S}$,

3. *for all* $S, R \in \mathcal{S}$ *such that* $S \not\subseteq R$ *and* $R \not\subseteq S$, *we have* $|V(R) \cap V(S)| \leq 1$.

The above definition recursively generates a cover. The splitting process that generates a hierarchical cover $\mathcal{S}$ of $T$ is formalized as rooted tree $(D, r)$ in the following definition.

**Definition 2.** *If* $\mathcal{S}$ *is a hierarchical cover of* $T$ *we define the associated **decomposition tree** $(D, r)$ as a rooted tree, whose vertex set* $V(D) := T^\bullet \cup E(T)$ *where* $D^\bullet = T^\bullet$ *and* $\mathrm{leaves}(D) = E(T)$, *such that the following three properties hold:*

1. $S^r = T$,

2. *for all* $c, p \in D^\bullet$, $c \in \downarrow_D^r(p)$ *iff* $S^c \in \Omega(S^p, p)$,

3. *for all* $(c, p) \in E(T)$ [1], *we have* $(c, p) \in \downarrow_D^r(p)$ *iff* $S^{(c,p)} \in \Omega(S^p, p)$.

The following lemma shows that with any given hierarchical cover $\mathcal{S}$ it is possible to associate a unique decomposition tree $(D, r)$.

**Lemma 3.** *A hierarchical cover* $\mathcal{S}$ *of* $T$ *defines a unique decomposition tree* $(D, r)$ *such that if* $S \in \mathcal{S}$ *there exists a* $v \in V(D)$ *such that* $S = S^v$ *and if* $v, w \in V(D)$ *and* $v \neq w$, *then* $S^v \neq S^w$.

For a given hierarchical cover $\mathcal{S}$ in the following we define the **height** and the **exposure**: two properties which measure different senses of the "size" of a cover. The **height** of a hierarchical cover $\mathcal{S}$ is the height of the associated decomposition tree $D$. Note that the height of a decomposition tree $D$ may be exponentially smaller than the height of $T$, since, for example, it is not difficult to show that there exists a decomposition tree isomorphic to a binary tree when the input tree $T$ is a path graph. If $R \subseteq T$ and $\mathcal{S}_R$ is a hierarchical cover of $R$, we define the **exposure** of $\mathcal{S}_R$ (with respect to tree $T$) as $\max_{Q \in \mathcal{S}_R} |\partial_T(Q)|$. Thus the exposure is a measure relative to a "containing" tree (which can be the input tree $T$ itself) and the height is independent of any containing tree.

In Section 4 the covering subtrees correspond to cached "joint distributions," which are defined on the boundary vertices of the subtrees, and require memory *exponential* in the boundary size. Thus we are interested in covers with small exposure.

We now define a measure of the optimal height with respect to a given exposure value.

**Definition 4.** *A hierarchical cover with exposure at most* $k$ *is called a* $k$***-hierarchical cover***. *Given any subtree* $R \subseteq T$, *the* $k$***-decomposition potential*** $\chi^k(R)$ *of* $R$ *is the minimum height of all hierarchical covers of* $\mathcal{S}_R$ *with exposure (with respect to* $T$) *not larger than* $k$. *The* $*$***-decomposition potential*** $\chi^*(R)$ *is the minimum height of all hierarchical covers of* $R$. *If* $|\partial_T(R)| > k$ *then* $\chi^k(R) := \infty$.

Let's consider some examples. Given a star graph, i.e., a graph with a single central vertex and any number of adjacent vertices, there is in fact only one possible hierarchical cover obtained by splitting the central vertex so that $\chi^*(\text{star}) = 1$. For path graphs, $\chi^*(\text{path}) = \Theta(\log |\text{path}|)$, as mentioned above. An interesting example is a star with path graphs rather than single edges. Specifically, a star-path may be formed by a set of $\frac{|\text{star-path}|}{\log |\text{star-path}|}$ path graphs $P_1, P_2, \ldots$ each with $\log |\text{star-path}|$ edges. These path graphs are then joined at a central vertex. In this case we have $\chi^*(\text{star-path}) = \mathcal{O}(\log \log(|\text{star-path}|))$; as each path has a hierarchical cover of height $\mathcal{O}(\log \log(|\text{star-path}|))$, each of these path covers may then be joined to create a cover of the star-path. In Theorem 6 we will see the generic bound $\chi^*(T) \leq \mathcal{O}(\min(\Delta(T), \log |V(T)|))$. The star-path thus illustrates that the bound may be exponentially loose.

In Theorem 6 we will see that $\chi^2(T) \leq 2\chi^*(T)$. Thus we may restrict our algorithm to hierarchical covers with an exposure of 2 at very little cost in efficiency. Hence, we will now focus our attention on 2-hierarchical covers.

**2-Hierarchical covers.** Given any element $Q \neq T$ in a 2-hierarchical cover of $T$ then $|\partial_T(Q)| \in \{1, 2\}$. Consider the case in which $\partial_T(Q) = \{v, w\}$, i.e. $|\partial_T(Q)| = 2$. Then $Q$ can be specified by

the two vertices $v, w$ and defined as follows: $Q := \begin{bmatrix} w \\ v \end{bmatrix} := \mathrm{argmax}_{S \subseteq T}(|V(S)| : v, w \in \mathrm{leaves}(S))$, that is the maximal subtree of $T$, having $v$ and $w$ among its leaves.

Consider now the case in which $\partial_T(Q) = \{w\}$, i.e. $|\partial_T(Q)| = 1$. $Q$ is now defined as the $T$'s subtree containing vertex $w$ together with all the descendents $\Downarrow_T^w(z)$ where $z \in N_T(w)$. Hence, a subtree such as $Q$ can be uniquely determined by the $w$'s neighbor $z \in N_T(w)$. In order to denote subtree $Q$ in this case we use the following notation: $Q := \begin{bmatrix} w \\ z \end{bmatrix}$. Observe that one can also represent a "boundary one" subtree with the previous notation by writing $Q := \begin{bmatrix} w \\ \ell \end{bmatrix}$, where $\ell$ is *any* [2] chosen leaf of $T$ belonging to $\Downarrow_T^w(z)$ (see Figure 1).

$(2, s)$-**Hierarchical covers.** We now introduce the notion of $(2, s)$-hierarchical covers (which, for simplicity, we shall also call $(2, s)$-covers) with respect to a rooted tree $(T, s)$. This notion explicitly depends on a given vertex $s \in V(T)$, which, for the sake of simplicity, will be assumed to be a leaf of $T$. $(2, s)$-Hierarchical covers are guaranteed to not be much larger than a 2-hierarchical cover (see Theorem 6). They are also amenable to a bottom-up construction.

**Definition 5.** *Given any subtree $R \subseteq T$, a 2-hierarchical cover $\mathcal{S}_R$ is a $(2, s)$-**hierarchical cover** of $R$ if, for all $S \in \mathcal{S}_R \setminus \{T\}$, there exists $v, w \in S$ where $v \in \Downarrow_T^s(w)$ such that (case 1: $|\partial_T(Q)| = 1$) $S = \begin{bmatrix} w \\ \bar{v} \end{bmatrix}$, or (case 2: $|\partial_T(Q)| = 2$) $S = \begin{bmatrix} w \\ v \end{bmatrix}$. In the former case $v \in \downarrow_T^s(w)$. We define $\chi_s^2(R)$ to be the **minimal height** of any possible $(2, s)$-hierarchical cover of $R \subseteq T$.*

Thus every subtree of a $(2, s)$-hierarchical cover is necessarily "oriented" with respect to a root $s$.

## 3  Computing an optimal hierarchical cover

From a "big picture" perspective, a $(2, s)$-hierarchical cover $\mathcal{G}$ is recursively constructed in a bottom-up fashion: in the initialization phase $\mathcal{G}$ contains only the atomic components convering $T$, i.e. the ones formed only by a pair of adjacent vertices of $V(T)$. We have then at this stage $|\mathcal{G}| = |E(T)|$. Then $\mathcal{G}$ grows step by step through the addition of new covering subtrees of $T$. At each time step $t$, at least one subtree of $T$ is added to $\mathcal{G}$. All the subtrees added at each step $t$ must strictly contain only subtrees added before step $t$.

We now introduce the formal description of our method for constructing a $(2, s)$-hierarchical cover $\mathcal{G}$. As we said, the construction of $\mathcal{G}$ proceeds in incremental steps. At each step $t$ the method operates on a tree $T_t$, whose vertices are part of $V(T)$. The construction of $T_t$ is accomplished starting by $T_{t-1}$ (if $t > 0$) in such a way that $V(T_t) \subset V(T_{t-1})$, where $T_0$ is set to be the subtree of $(T, s)$ containing the root and all the internal vertices.

During each step $t$ all the while-loop instructions of Figure 1 are executed: (1) some vertices (the black ones in Figure 1) are selected through a depth-first visit (during the backtracking steps) of $T_t$ starting from $s$ [3], (2) for each selected vertex $v$, subtree $S^v$ is obtained from merging subtrees added to $\mathcal{G}$ in previous steps and overlapping at vertex $v$, (3) in order to create tree $T_{t+1}$ from $T_t$ the previously selected vertices of $T_t$ are removed, (4) the edge set $E(T_{t+1})$ is created from $E(T_t)$ in such a way to preserve the $T_t$'s structure, but all the edges incident to the vertices removed from $V(T_t)$ (the black vertices Figure 1) in the while-loop step 3 need to be deleted. The possible disconnection that would arise by the removal of these parts of $T_t$ is avoided by completing the construction of $E_{t+1}$ through the addition of some new edges. These additional edges are not part of $E(T)$ and link each vertex $v$ with its grand-parent in $T_t$ if vertex $v$'s parent was deleted (see the dashed line edges in Figure 1) during the construction of $T_{t+1}$ from $T_t$. In the final while-loop step the variable $t$ gets incremented by 1.

Basically, the key for obtaining optimality with this construction method can be explained with the following observation. At each time step $t$, when we add a covering subtree $S^v$ for some vertex $v \in V(T_t)$ selected by the algorithm (black vertices in Figure 1), the whole $(2, s)$-cover of $S^v$ becomes completely contained in $\mathcal{G}$ and its height is $t + 1$, which can be proven to be the minimum possible height of a $(2, s)$-cover of $S^v$. Hence, at each time step $t$ we construct the $t + 1$-th level (in the hierarchical nested sense) of $\mathcal{G}$ in such a way to achieve local optimality for all elements contained in all levels smaller or equal to $t + 1$. As the next theorem states, the running of the algorithm is linear in $|V(T)|$.

**Theorem 6.** *Given a rooted tree $(T, s)$, the algorithm in Figure 1 outputs $\mathcal{G}$, an optimal $(2, s)$-hierarchical cover in time linear in $|V(T)|$ of height $\chi_s^2(T)$ which is bounded as $\chi^*(T) \leq \chi^2(T) \leq \chi_s^2(T) \leq 2\chi^*(T) \leq \mathcal{O}(\min(\log|V(T)|, \Delta(T)))$.*

Before we provide the detailed description of the algorithm for constructing an optimal $(2, s)$-hierarchical cover we need some ancillary definitions. We call a vertex $v \in V(T_t) \setminus \{s\}$ **mergeable** (at time $t$) if and only if either (i) $v \in \text{leaves}(T_t)$ or (ii) $v$ has a single child in $T_t$ and that child is not mergeable. If $v \in V(T_t) \setminus \{s\}$ is mergeable we write $v \in \mathcal{M}_t$. We also use the following shorthands for making more intuitive our notation: We set $c_v^t := \downarrow_{T_t}^s(v)$ when $|\downarrow_{T_t}^s(v)| = 1$, $p_v^t := \uparrow_{T_t}^s(v)$ when $v \neq s$ and $g_v^t := \uparrow_{T_t}^s(p_v^t)$ when $v, p_v^t \neq s$. Finally, given $u, u' \in V(T)$ such that $u' \in \Downarrow_T^s(u)$, we indicate with $\downarrow_T^s(u \mapsto u')$ the child of $u$ which is ancestor of $u'$ in $T$.

---

**Input:** Rooted tree $(T, s)$.

---

**Initialisation:** $T_0 \leftarrow T^\bullet \cup \{s\}$; $\quad t \leftarrow 0$;
$\qquad \mathcal{G} \leftarrow \left\{ \left[ \uparrow_v^{s(v)} \right] : v \in V(T) \setminus \{s\} \right\}$.

---

**While** $\left( V(T_t) \neq \{s\} \right)$

    1. Construct $\mathcal{M}_t$ via depth-first search of $T_t$ from $s$.

    2. **For** all $v \in \mathcal{M}_t$, merge as follows:
        **If** $v \in \text{leaves}(T_t)$ **then**
           $z \leftarrow \downarrow_T^s(p_v^t \mapsto v)$.

           $\mathcal{G} \leftarrow \mathcal{G} \cup \left[ \begin{smallmatrix} p_v^t \\ z \end{smallmatrix} \right]$.

        **Else** $\mathcal{G} \leftarrow \mathcal{G} \cup \left[ \begin{smallmatrix} p_v^t \\ c_v^t \end{smallmatrix} \right]$.

    3. $V(T_{t+1}) \leftarrow V(T_t) \setminus \mathcal{M}_t$.

    4. $E(T_{t+1}) \leftarrow \{(v, p_v^t) : v, p_v^t \in V(T_{t+1})\} \cup$
              $\{(v, g_v^t) : v, g_v^t \in V(T_{t+1}),$
              $p_v^t \notin V(T_{t+1})\}$.

    5. $t \leftarrow t + 1$.

---

**Output:** Optimal $(2, s)$-hierarchical cover $\mathcal{G}$ of $T$.

---

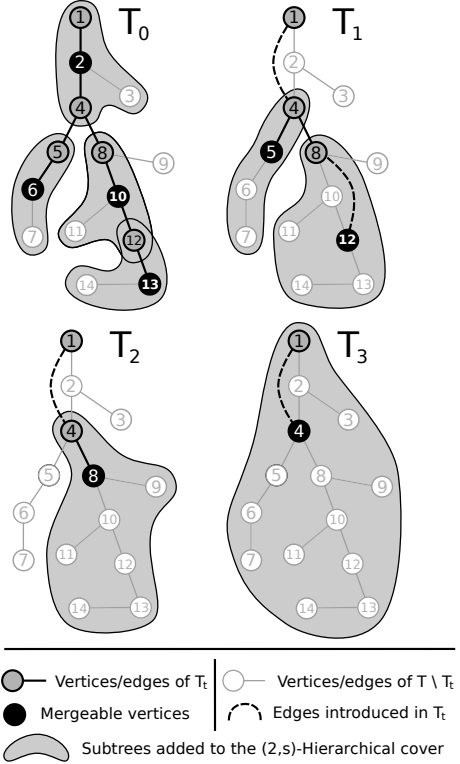

Vertices/edges of $T_t$    Vertices/edges of $T \setminus T_t$
Mergeable vertices    Edges introduced in $T_t$
Subtrees added to the (2,s)-Hierarchical cover

**Figure 1: Left**: Pseudocode for the linear time construction algorithm for an optimal $(2, s)$-hierarchical cover. **Right**: Pictorial explanation of the pseudocode and the details of the hierarchical cover.
In order to clarify the method, we describe some of the details of the cover and some merge operations that are performed in the diagram. Vertex 1 is the root vertex $s$. In each component, depicted as enclosed in a line, the black node is the splitting vertex, i.e., a mergeable vertex of the tree $T_t$. The boundary definition may be clarified by highlighting, for instance, that $\partial_T(S^2) = \{4\}$ and $\partial_T(S^{10}) = \{8, 12\}$. Subtree $S^2$ contains vertices 1, 2, 3 and 4. Vertex 2 is the splitting vertex of $S^2$. $\Omega(S^2, 2) = \{S^{(1,2)}, S^{(2,3)}, S^{(2,4)}\}$, i.e., at time $t = 0$, $S^2$ is formed by merging the three atomic subtrees $S^{(1,2)}$, $S^{(2,3)}$ and $S^{(2,4)}$, which were added in the initialization step. These three subtrees overlap at only vertex 2, which is depicted in black because it is mergeable in $T_0$. For what concerns the decomposition tree $(D, r)$, we have $\downarrow_D^r(5) = \{(4, 5), 6\}$, which implies that $S^5$ is therefore formed by the atomic component $S^{(4,5)}$ and the non-atomic component $S^6$. At time $t = 1$, $S^{12}$ is obtained by merging $S^{10}$ together with $S^{13}$, which have been both created at time $t = 0$. Observe that in $T_1$ vertex 12 is a leaf and the $z$ variable in the while-loop step 2 is assigned to vertex 10 ($v$ and and $p_v^t$ is respectively vertex 12 and 8). Regarding the subtree representation with the square bracket notation we can write, for instance, $S^2 = \left[ \begin{smallmatrix} 1 \\ 4 \end{smallmatrix} \right]$ and $S^{12} = \left[ \begin{smallmatrix} 8 \\ 10 \end{smallmatrix} \right] \left( \equiv \left[ \begin{smallmatrix} 8 \\ 11 \end{smallmatrix} \right] \equiv \left[ \begin{smallmatrix} 8 \\ 14 \end{smallmatrix} \right] \right)$. Observe that, according to the definition of a $(2, s)$-hierarchical cover, we have $4 \in \Downarrow_T^1(1)$ and $10 \in \downarrow_T^1(8)$. Finally, notice that the height of the $(2, s)$-hierarchical cover of $S^v$ is equal to $t + 1$ iff $v$ is depicted in black in $T_t$.

## 4 Online marginalization

In this section we introduce our algorithm for efficiently computing marginals by summing over products of variables in a tree topology. Formally our model is specified by a triple $(T, \Theta, \mathcal{D})$ where

$T$ is a tree, $\Theta = (\theta_{e,l,m} : e \in E(T), l \in \mathbb{N}_k, m \in \mathbb{N}_k)$ so that $\theta_e$ is a positive symmetric $k \times k$ matrix and $\mathcal{D} = (d_{v,c} : v \in V(T), c \in \mathbb{N}_k)$ is a $|V(T)| \times k$ matrix. In a probabilistic setting it is natural to view each normalized $\theta_e$ as a stochastic symmetric "transition" matrix and the "data" $\mathcal{D}$ as a right stochastic matrix corresponding to "beliefs" about $k$ different labels at each vertex in $T$. In our online setting $\Theta$ is a fixed parameter and $\mathcal{D}$ is changing over time and thus an element in a sequence $(\mathcal{D}^1, \ldots, \mathcal{D}^t, \ldots)$ where successive elements only differ in a single row. Thus at each point at time we receive information at a single vertex.

In our intended application (see Section 5) of the model there is no necessary "randomness" in the generation of the data. However the language of probability provides a natural metaphor we use for our computed quantities. Thus a ($k$-ary) labeling of $T$ is a vector $\mu \in \mathcal{L}$ with $\mathcal{L} := \mathbb{N}_k^{V(T)}$ and its "probability" with respect to $(\Theta, \mathcal{D})$ is

$$p(\mu|\Theta, \mathcal{D}) := \frac{1}{Z} \prod_{(i,j) \in E(T)} \theta_{(i,j),\mu(i),\mu(j)} \prod_{v \in V(T)} d_{v,\mu(v)}, \tag{1}$$

with the normalising constant $Z := \sum_{\mu \in \mathcal{L}} \prod_{(i,j) \in E(T)} \theta_{(i,j),\mu(i),\mu(j)} \prod_{v \in V(T)} d_{v,\mu(v)}$. We denote the marginal probability at a vertex $v$ as

$$p(v \to a|\Theta, \mathcal{D}) := \sum_{\mu \in \mathcal{L} \,:\, \mu(v)=a} p(\mu|\Theta, \mathcal{D}). \tag{2}$$

**Using the hierarchical cover for efficient online marginalization.** In the previous section we discussed a method to compute a hierarchical cover of a tree $T$ with optimal height $\chi_s^2(T)$ in time linear in $T$. In this subsection we will show how these covering components form a covering set of cached "marginals'". So that we may either compute $p(v \to a|\Theta, \mathcal{D})$ or update a single row of the data matrix $\mathcal{D}$ and recompute the changed cached marginals all in time linear in $\chi_s^2(T)$.

**Definition 7.** *Given a tree $S \subseteq T$, the **potential** function, $\psi_T^S : \mathcal{L}(\partial_T(S)) \to \mathbb{R}$ with respect to $(\Theta, \mathcal{D})$ is defined by:*

$$\psi_T^S(\tilde{\mu}) := \sum_{\mu \in \mathcal{L}(S) \,:\, \mu(\partial_T(S))=\tilde{\mu}} \left( \prod_{(v,w) \in E(S)} \theta_{(v,w),\mu(v),\mu(w)} \right) \left( \prod_{v \in S \setminus \partial_T(S)} d_{v,\mu(v)} \right) \tag{3}$$

Where $\mathcal{L}(X) := \mathbb{N}_k^X$ with $X \subseteq V(T)$ is thus the restriction of $\mathcal{L}$ to $X$ and likewise if $\mu \in \mathcal{L}$ then $\mu(X) \in \mathcal{L}(X)$ is the restriction of $\mu$ to $X$. For each tree in our hierarchical cover $S \in \mathcal{S}$ we will have an associated potential function. Intuitively each of these potential functions summarize the information in their interior by the marginal function defined on their boundary. Thus trees $S \in \mathcal{S}$ with a boundary size of 1 require $k$ values to be cached, the "$\alpha$" weights; while boundary size 2 trees requires $k^2$ values, the "$\beta$" weights. This clarifies our motivation to find a cover with both small height and exposure. We also cache $\gamma$ weights that represent the product of $\alpha$ weights; these weights allow efficient computation on high degree vertices. The set of cached values necessary for fast online computation correspond to these three types of weights of which there is a linear quantity and on any given update or marginalization step only $O(\chi_s^2(T))$ of them are accessed.

**Definitions of weights and potentials.** Given a tree $T$ and a hierarchical cover $\mathcal{S}$ it is isomorphic to a decomposition tree $(D, r)$. The decomposition tree will serve a dual purpose. First, each vertex $z \in D$ will serve as a "name" for a tree $S^z \in \mathcal{S}$. Second, in the same way that the "messages passing" in belief propagation the follows the topology of the input tree, the structure of our computations follows the decomposition tree $D$. We now introduce our notations for computing and traversing the decomposition tree. As the cover has trees with one or two boundary vertices (excepting $T$ which has none) we define the corresponding vertices of the decomposition tree, $C_i := \{z \in D : |\partial_T(S^z)| = i\}$ for $i \in \{1, 2\}$. In this section since we are concerned with the traversal of $(D, r)$ we abbreviate $\downarrow_D, \uparrow_D$ as both $\downarrow, \uparrow$ respectively as convenient. As $\downarrow_D(v)$ is a set of children, we define the following functions to select specific children, $\triangleleft(v) := w$ if $w \in \downarrow(v), \uparrow(v) \in \partial_T(S^{(w)})$ for $v \in D^\bullet \cap (C_1 \cup C_2)$ and $\triangleright(v) := w$ if $w \in \downarrow(v), w \neq \triangleleft(v)$ for $w \in C_2$ and $v \in D^\bullet \cap C_2$. When clear from the context we will use $\triangleleft v$ for $\triangleleft(v)$ as well as $\triangleright v$ for $\triangleright(v)$. We also need notation for the potentially two boundary vertices of a tree $S^v \in \mathcal{S}$ if $v \in D \setminus \{r\}$. Observe that for $v \in C_1 \cup C_2$ one boundary vertex of $S^v$ is necessarily $\dot{v} := \uparrow v$ and if $v \in C_2$ there exists an ancestor $\ddot{v}$ of $v$ in $D$ of so that $\{\dot{v}, \ddot{v}\} = \partial_T(S^v)$. We also extend the split notation to pick out the specific

| | | | |
|---|---|---|---|
| $\alpha_a(v) := \psi_T^{S^v}(\dot{v} \to a),$ | $(v \in C_1)$ | $\gamma_a(v) := d_{va} \prod\limits_{w \in \downarrow(v) \cap C_1} \alpha_a(w),$ | $(v \in V(T))$ |
| $\beta_{ab}(v) := \psi_T^{S^v}(\dot{v} \to a, \ddot{v} \to b),$ | $(v \in C_2)$ | $\rho_a(v) := d_{va} \prod\limits_{R \in \Omega(T,v)} \psi_T^R(v \to a),$ | $(v \in V(T))$ |
| $\delta_a^{\triangleleft}(v) := d_{\dot{v}a}\, \psi_T^{\overline{\Omega(T,\dot{v},v)}}(\dot{v} \to a),$ | $(v \in V(T) \setminus \{r\})$ | $\delta_a^{\triangleright}(v) := d_{\ddot{v}a}\, \psi_T^{\overline{\Omega(T,\ddot{v},v)}}(\ddot{v} \to a),$ | $(v \in C_2)$ |
| $\epsilon_a^{\triangleleft}(v) := \psi_T^{\Omega(T,v,\dot{v})}(v \to a),$ | $(v \in V(T) \setminus \{r\})$ | $\epsilon_a^{\triangleright}(v) := \psi_T^{\Omega(T,v,\ddot{v})}(v \to a),$ | $(v \in C_2)$ |

Table 1: Weight definitions

complementary subtrees of $T$ resulting from a split thus $\Omega(T,p,q) := Q \in \Omega(T,p)$ if $q \in Q$ and define $\overline{\Omega(T,p,q)} := \cup\{R \in \Omega(T,p) : q \notin R\}$. Observe that $T = \Omega(T,p,q) \cup \overline{\Omega(T,p,q)}$ and $\{p\} = \Omega(T,p,q) \cap \overline{\Omega(T,p,q)}$. We shall use the notation $(v_1 \to a_1, v_2 \to a_2, \ldots, v_m \to a_m)$ to represent a labeling of $\{v_1, v_2, \ldots, v_m\}$ that maps $v_i$ to $a_i$. In Table 1 we now give the weights used in our online marginalization algorithm. The $\alpha_a, \beta_{ab}, \gamma_a$ weights are cached values maintained by the algorithm and the weights $\rho_a, \delta_a^{\triangleleft}, \delta_a^{\triangleright}, \epsilon_a^{\triangleleft}$, and $\epsilon_a^{\triangleright}$ are temporary values[4] computed "on-the-fly." The indices $a, b \in \mathbb{N}_k$ and thus the memory requirements of our algorithm are linear in the cardinality of the tree and quadratic in the number of labels.

**Identities for weights and potentials.** For the following lemma we introduce the notion of the extension of a labelling. We extend by a vertex $v \in V(T)$ and a label $a \in \mathbb{N}_k$, the labelling $\mu \in \mathcal{L}(X)$ to the labelling $\mu_v^a \in \mathcal{L}(X \cup \{v\})$ which satisfies $\mu_v^a(v) = a$ and $\mu_v^a(X) = \mu$.

**Lemma 8.** *Given a tree, $S \subseteq T$, and a vertex $v \in S$ then if $v \in S \setminus \partial_T(S)$*

$$\psi_T^S(\mu) = \sum_{a \in \mathbb{N}_k} d_{va} \prod_{R \in \Omega(S,v)} \psi_T^R(\mu_v^a(\partial_T(R))) \text{ else if } v \in \partial_T(S) \text{ then } \psi_T^S(\mu) = \prod_{R \in \Omega(S,v)} \psi_T^R(\mu(\partial_T(R)))$$

Thus a direct consequence of Lemma 8 is that we can compute the marginal probability at $v$ as $p(v \to a | \Theta, \mathcal{D}) = \dfrac{\rho_a(v)}{\sum_{b \in \mathbb{N}_k} \rho_b(v)}$ . The recursive application of such factorizations is the basis of our algorithm (these factorizations are summarized in Table 2 in the technical appendices).

**Algorithm initialization and complexity.** In Figure 2 we give our algorithm for computing the marginals at vertices with respect to $(\Theta, \mathcal{D})$. A number of our identities assumed for a given vertex that it is in the interior of the tree and hence in the interior of decomposition tree. Thus before we find the hierarchical cover of our input tree we extend the tree by adding a "dummy" edge from each leaf of the tree to a new dummy vertex. These dummy edges play no role except to simplify notation. The hierarchical cover is then found on this enlarged tree; the cover height may at most only increase by one. By setting the values in dummy edges and vertices in $\Theta$ and $\mathcal{D}$ to one, this ensures that all marginal computations are unchanged.

The running time of the algorithm is as follows. The computation of the hierarchical cover[5] is linear in $|V(T)|$ as is the initialization step. The update and marginalization are linear in cover height $\chi^*(T)$. The algorithm also scales quadratically in $k$ on the marginalization step and cubically in $k$ on update as the merge of two $C_2$ trees require the multiplication of two $k \times k$ matrices. Thus for example if the set of possible labels is linear in the size of the tree classical belief propagation may be faster.

Finally we observe that we may reduce the cubic dependence to a quadratic dependence on $k$ via a cover with the height bounded by the diameter of $T$ as opposed to $\chi^*(T)$. This follows as the only cubic step is in the update of a non-atomic (non-edge) $\beta$-potential. Thus if we can build a cover, with only atomic $\beta$-potentials the running time will scale with $k$ quadratically. We accomplish this by modifying the cover algorithm (Figure 1) to only merge leaf vertices. Observe that the height of this cover is now $\mathcal{O}(\text{diameter}(T))$; and we have a hierarchical factorization into $\alpha$-potentials and only atomic $\beta$-potentials.

## 5   Multi-task learning in the allocation model with TREE-HEDGE

We conclude by sketching a simple online learning application to multi-task learning that is amenable to our methods. The inspiration is that we have multiple tasks and a *given* tree structure that describes our prior expectation of "relatedness" between tasks (see e.g., [7, Sec. 3.1.3]).

**Marginalization** (*vertex* $v \in D^\bullet$) :

1.    $w \leftarrow r$
2.    $\rho_a(w) \leftarrow \gamma_a(r)$
3.    **while**($w \neq v$)
4.      $w \leftarrow \uparrow^v(w)$
5.      **if**($w \in C_1$)
6.        $\delta_a^\lhd(w) \leftarrow \rho_a(\uparrow(w))/\alpha_a(w)$
7.        $\epsilon_a^\lhd(w) \leftarrow \sum_b \beta_{ab}(\lhd(w))\delta_b^\lhd(w)$
8.        $\rho_a(w) = \gamma_a(w)\epsilon_a^\lhd(w)$
9.      **else**
10.        **if**($w = \lhd(\uparrow(w))$)
11.          $\delta_a^\lhd(w) \leftarrow \epsilon_a^\rhd(\uparrow(w))\gamma_a(\uparrow(w))$
12.          $\delta_a^\rhd(w) \leftarrow \delta_a^\lhd(\uparrow(w))$
13.        **else**
14.          $\delta_a^\lhd(w) \leftarrow \epsilon_a^\lhd(\uparrow(w))\gamma_a(\uparrow(w))$
15.          $\delta_a^\rhd(w) \leftarrow \delta_a^\rhd(\uparrow(w))$
16.        $\epsilon_a^\lhd(w) \leftarrow \sum_b \delta_b^\lhd(w)\beta_{ab}(\lhd(w))$
17.        $\epsilon_a^\rhd(w) \leftarrow \sum_b \delta_b^\rhd(w)\beta_{ab}(\rhd(w))$
18.        $\rho_a(w) \leftarrow \epsilon_a^\lhd(w)\epsilon_a^\rhd(w)\gamma_a(w)$
19.
20. **Output:** $\rho_a(v)/(\sum_b \rho_b(v))$

**Initialization:** The $\alpha$, $\beta$ and $\gamma$ weights are initialised in a bottom-up fashion on the decomposition tree - we initialise the weights of a vertex after we have initialised the weights of all its children. Specifically, we first do a depth-first search of $D$ starting from $r$: When we reach an edge $(v, w) \in E(T)$, if neither $v$ or $w$ is a leaf then we set $\beta_{ab}((v,w)) \leftarrow \theta_{(v,w),a,b}$ otherwise assuming $w$ is a leaf we set $\alpha_a(v) \leftarrow 1$ (dummy edge). When we reach a vertex, $v \in V(T)$, for the last time (i.e. just before we backtrack from $v$) then set: $\gamma_a(v) \leftarrow d_{va} \prod_{w \in \downarrow(v) \cap C_1} \alpha_a(w)$, and if $v \in C_2$ then $\beta_{ab}(v) \leftarrow \sum_c \beta_{ca}(\lhd(v))\beta_{cb}(\rhd(v))\gamma_c(v)$, or if $v \in C_1$ then $\alpha_a(v) \leftarrow \sum_c \beta_{ca}(\lhd(v))\gamma_c(v)$.

**Update** (*vertex* $v \in D^\bullet$ ; *data* $d \in [0, \infty)^k$):

1.    $\gamma_a(v) \leftarrow \gamma_a(v)\frac{d_a}{d_{va}}$ ; $d_v \leftarrow d; w \leftarrow v$
2.    **while**($w \neq r$)
3.      **if**($w \in C_1$)
4.        $\alpha_a^{\mathrm{old}} \leftarrow \alpha_a(w)$
5.        $\alpha_a(w) \leftarrow \sum_c \beta_{ca}(\lhd(w))\gamma_c(w)$
6.        $\gamma_a(\uparrow(w)) \leftarrow \gamma_a(\uparrow(w))\alpha_a(w)/\alpha_a^{\mathrm{old}}$
7.      **else**
8.        $\beta_{ab}(w) \leftarrow \sum_c \beta_{ca}(\lhd(w))\beta_{cb}(\rhd(w))\gamma_c(w)$
9.      $w \leftarrow \uparrow(w)$

Figure 2: Algorithm: Initialization, Marginalization and Update

1. **Parameters:** A triple $(T, \Theta, \mathcal{D}^1)$ and $\eta \in (0, \infty)$.
2. For $t = 1$ to $\ell$ do
3.    Receive: $v^t \in V(T)$
4.    Predict: $\hat{p}^t = (p(v^t \rightarrow a|\Theta, \mathcal{D}^t))_{a \in \mathbb{N}_k}$
5.    Receive: $y^t \in [0, 1]^k$
6.    Incur loss: $L_{\mathrm{mix}}(y^t, \hat{p}^t)$
7.    Update: $\mathcal{D}^{t+1} = \mathcal{D}^t$ ; $\mathcal{D}^{t+1}(v^t) = (\hat{p}^t(a)e^{-\eta y^t(a)})_{a \in \mathbb{N}_k}$

Figure 3: TREE-HEDGE

Thus each vertex represents a task and if we have an edge between vertices then *a priori* we expect those tasks to be related. Thus the hope is that information received for one task (vertex) will allow us to improve our predictions on another task. For us each of these tasks is an *allocation task* as addressed often with the HEDGE algorithm [4]. A similar application of the HEDGE algorithm in multi-task learning was given in [8]. Their the authors considered a more challenging set-up where the task structure is unknown and the hope is to do well if there is *a posteriori* a small clique of closely related tasks. Our strong assumption of prior "tree-structured" knowledge allows us to obtain a very efficient algorithm and sharp bounds which are not directly comparable to their results. Finally, this set-up is also closely related to online graph labeling problem as in e.g., [9, 10, 11].

Thus the set-up is as follows. We incorporate our prior knowledge of task-relatedness with the triple $(T, \Theta, \mathcal{D}^1)$. Then on a trial $t$, the algorithm is given a $v^t \in V(T)$, representing the task. The algorithm then gives a non-negative prediction vector $\hat{p}^t \in \{p : \sum_{a=1}^k p(a) = 1\}$ for task $v^t$ and receives an outcome $y^t \in [0, 1]^k$. It then suffers a mixture loss $L_{\mathrm{mix}}(y^t, \hat{p}^t) := y^t \cdot \hat{p}^t$. The aim is to predict to minimize this loss. We give the algorithm in Figure 3. The notation follows Section 4 and the method therein implies that on each trial we can predict and update in $\mathcal{O}(\chi^*(T))$ time. We obtain the following theorem (a proof sketch is contained in appendix C of the long version).

**Theorem 9.** *Given a tree $T$, a vertex sequence $\langle v^1, \ldots, v^\ell \rangle$ and an outcome sequence $\langle y^1, \ldots, y^\ell \rangle$ the loss of the* TREE-HEDGE *algorithm with the parameters $(\Theta, \mathcal{D}^1)$ and $\eta > 0$ is, for all labelings $\mu \in \mathbb{N}_k^{V(T)}$, bounded by*

$$\sum_{t=1}^\ell L_{mix}(y^t, \hat{p}^t) \leq c_\eta \left( \sum_{t=1}^\ell y^t(\mu(v^t)) + \frac{\ln 2}{\eta} \frac{1}{\log_2 p(\mu|\Theta, \mathcal{D}^1)} \right) \quad \text{with } c_\eta := \frac{\eta}{1 - e^{-\eta}} . \quad (4)$$

**Acknowledgements.** We would like to thank David Barber, Guy Lever and Massimiliano Pontil for valuable discussions. We, also, acknowledge the financial support of the PASCAL 2 European Network of Excellence.

## Footnotes

[1] Observe that $(c, p) \in E(T)$ implies $c, p \in V(T)$ and $(c, p) \in \mathrm{leaves}(D)$.

[2]This representation is not necessarily unique, as if $\ell_1, \ell_2 \in \mathrm{leaves}(T) \cap Q$, we have $\begin{bmatrix} w \\ \ell_1 \end{bmatrix} = \begin{bmatrix} w \\ \ell_2 \end{bmatrix} \left(= \begin{bmatrix} w \\ \bar{z} \end{bmatrix}\right)$.

[3]Observe that $s$ is the unique vertex belonging to $V(T_t)$ for all time steps $t \geq 0$.

[4]Note: if for $\gamma_a(v)$ if the product is empty then the product evaluates to 1; and if $v \in C_1$ then $\epsilon_a^{\triangleright}(v) := 1$.

[5]The construction of the decomposition tree may be simultaneously accomplished with the same complexity.

# References

[1] David Barber. *Bayesian Reasoning and Machine Learning*. Cambridge University Press, 2012.

[2] Christopher M. Bishop. *Pattern Recognition and Machine Learning*. Springer, 2006.

[3] Frank R. Kschischang, Brenden J. Frey, and Hans Andrea Loeliger. Factor graphs and the sum-product algorithm. *IEEE Transactions on Information Theory*, 47(2):498–519, 2001.

[4] Yoav Freund and Robert E Schapire. A decision-theoretic generalization of on-line learning and an application to boosting. *Journal of Computer and System Sciences*, 55(1):119–139, 1997.

[5] Judea Pearl. Reverend Bayes on inference engines: A distributed hierarchical approach. In *Proc. Natl. Conf. on AI*, pages 133–136, 1982.

[6] Arthur L. Delcher, Adam J. Grove, Simon Kasif, and Judea Pearl. Logarithmic-time updates and queries in probabilistic networks. *J. Artif. Int. Res.*, 4:37–59, February 1996.

[7] Theodoros Evgeniou, Charles A. Micchelli, and Massimiliano Pontil. Learning multiple tasks with kernel methods. *Journal of Machine Learning Research*, 6:615–637, 2005.

[8] Jacob Abernethy, Peter L. Bartlett, and Alexander Rakhlin. Multitask learning with expert advice. In *COLT*, pages 484–498, 2007.

[9] Mark Herbster, Massimiliano Pontil, and Lisa Wainer. Online learning over graphs. In *ICML*, pages 305–312. ACM, 2005.

[10] Mark Herbster, Guy Lever, and Massimiliano Pontil. Online prediction on large diameter graphs. In *NIPS*, pages 649–656. MIT Press, 2008.

[11] Nicolò Cesa-Bianchi, Claudio Gentile, and Fabio Vitale. Fast and optimal prediction on a labeled tree. In *COLT*, 2009.

